# Robust design of biological experiments

**Patrick Flaherty**
EECS Department
University of California
Berkeley, CA 94720
*flaherty@berkeley.edu*

**Michael I. Jordan**
Computer Science and Statistics
University of California
Berkeley, CA 94720
*jordan@cs.berkeley.edu*

**Adam P. Arkin**
Bioengineering Department,
LBL, Howard Hughes Medical Institute
University of California
Berkeley, CA 94720
*aparkin@lbl.gov*

## Abstract

We address the problem of robust, computationally-efficient design of biological experiments. Classical optimal experiment design methods have not been widely adopted in biological practice, in part because the resulting designs can be very brittle if the nominal parameter estimates for the model are poor, and in part because of computational constraints. We present a method for robust experiment design based on a semidefinite programming relaxation. We present an application of this method to the design of experiments for a complex calcium signal transduction pathway, where we have found that the parameter estimates obtained from the robust design are better than those obtained from an "optimal" design.

## 1   Introduction

Statistical machine learning methods are making increasing inroads in the area of biological data analysis, particularly in the context of genome-scale data, where computational efficiency is paramount. Learning methods are particularly valuable for their ability to fuse multiple sources of information, aiding the biologist to interpret a phenomenon in its appropriate cellular, genetic and evolutionary context. At least as important to the biologist, however, is to use the results of data analysis to aid in the design of further experiments. In this paper we take up this challenge—we show how recent developments in computationally-efficient optimization can be brought to bear on the problem of the design of experiments for complex biological data. We present results for a specific model of calcium signal transduction in which choices must be made among 17 kinds of RNAi knockdown experiments.

There are three main objectives for experiment design: parameter estimation, hypothesis testing and prediction. Our focus in this paper is parameter estimation, specifically in the setting of nonlinear kinetic models [1]. Suppose in particular that we have a nonlinear

model $y = f(x, \theta) + \varepsilon, \varepsilon \sim \mathcal{N}(0, \sigma^2)$, where $x \in \mathcal{X}$ represents the controllable conditions of the experiment (such as dose or temperature), $y$ is the experimental measurement and $\theta \in \mathbb{R}^p$ is the set of parameters to be estimated. We consider a finite menu of available experiments $\mathcal{X} = \{x_1, \ldots, x_m\}$. Our objective is to select the best set of $N$ experiments (with repeats) from the menu. Relaxing the problem to a continuous representation, we solve for a distribution over the design points and then multiply the weights by $N$ at the end [2]. The experiment design is thus

$$\xi = \left\{ \begin{array}{c} x_1, \ldots, x_m \\ w_1, \ldots, w_m \end{array} \right\}, \quad \sum_{i=1}^{m} w_i = 1, \quad w_i \geq 0, \forall i, \tag{1}$$

and it is our goal to select values of $w_i$ that satisfy an experimental design criterion.

## 2 Background

We adopt a standard least-squares framework for parameter estimation. In the nonlinear setting this is done by making a Taylor series expansion of the model about an estimate $\theta_0$ [3]

$$f(x, \theta) \approx f(x, \theta_0) + V(\theta - \theta_0), \tag{2}$$

where $V$ is the Jacobian matrix of the model; the $i^{th}$ row of $V$ is $v_i^T = \frac{\partial f(x_i, \theta)}{\partial \theta}\Big|_{\theta_0}$.

The least-squares estimate of $\theta$ is $\hat{\theta} = \theta_0 + \left(V^T W V\right)^{-1} V^T W \left(y - f(x, \theta_0)\right)$, where $W = \text{diag}(w)$. The covariance matrix for the parameter estimate is $\text{cov}(\hat{\theta}|\xi) = \sigma^2 \left(V^T W V\right)^{-1}$, which is the inverse of the observed Fisher information matrix.

The aim of optimal experiment design methods is to minimize the covariance matrix of the parameter estimate [4, 5, 6]. There are two well-known difficulties that must be surmounted in the case of nonlinear models [6]:

- The optimal design depends on an evaluation of the derivative of the model with respect to the parameters at a particular parameter estimate. Given that our goal is parameter estimation, this involves a certain circularity.

- Simple optimal design procedures tend to concentrate experimental weight on only a few design points [7]. Such designs are overly optimistic about the appropriateness of the model, and provide little information about possible lack of fit over a wider experimental range.

There have been three main responses to these problems: sequential experiment design [7], Bayesian methods [8], and maximin approaches [9].

In the sequential approach, a working parameter estimate is first used to construct a tentative experiment design. Data are collected under that design and the parameter estimate is updated. The procedure is iterated in stages. While heuristically reasonable, this approach is often inapplicable in practice because of costs associated with experiment set-up time.

In the Bayesian approach exemplified by [8], a proper prior distribution is constructed for the parameters to be estimated. The objective function is the KL divergence between the prior distribution and the expected posterior distribution; this KL divergence is *maximized* (thereby maximizing the amount of expected information in the experiment design). Sensitivity to priors is a serious concern, however, particularly in the biological setting in which it can be quite difficult to choose priors for quantities such as bulk rates for a complex process.

The maximin approach considers a bounded range for each parameter and finds the optimal design for the worst case parameters in that range. The major difficulties with this approach are computational, and its main applications have been to specialized problems [7].

The approach that we present here is closest in spirit to the maximin approach. We view both of the problems discussed above as arguments for a *robust* design, one which is insensitive to the linearization point and to model error. We work within the framework of E-optimal design (see below) and consider perturbations to the rank-one Fisher information matrix for each design point. An optimization with respect to such perturbations yields a robust semidefinite program [10, 11, 12].

## 3 Optimal Experiment Design

The three most common scalar measures of the size of the parameter covariance matrix in optimal experiment design are:

- *D-optimal design*: determinant of the covariance matrix.
- *A-optimal design*: trace of the covariance matrix.
- *E-optimal design*: maximum eigenvalue of the covariance matrix.

We adopt the E-optimal design criterion, and formulate the design problem as follows:

$$\mathcal{P}_0 : p_0^* = \min_w \lambda_{\max} \left[ \left( \sum_{i=1}^m w_i v_i v_i^T \right)^{-1} \right] \quad s.t. \quad \sum_{i=1}^m w_i = 1 \tag{3}$$
$$w_i \geq 0, \forall i,$$

where $\lambda_{\max}[M]$ is the maximum eigenvalue of a matrix $M$. This problem can be recast as the following semidefinite program [5]:

$$\mathcal{P}_0 : p_0^* = \max_{w,s} s \quad s.t. \quad \sum_{i=1}^m w_i v_i v_i^T \geq sI_p \tag{4}$$
$$\sum_{i=1}^m w_i = 1, \quad w_i \geq 0, \forall i,$$

which forms the basis of the robust extension that we develop in the following section.

## 4 Robust Experiment Design

The uncertain parameters appear in the experiment design optimization problem through the Jacobian matrix, $V$. We consider additive unstructured perturbations on the Jacobian or "data" in this problem. The uncertain observed Fisher information matrix is $F(w, \Delta) = \sum_{i=1}^m w_i(v_i v_i^T - \Delta_i)$, where $\Delta_i$ is a $p \times p$ matrix for $i = 1, \ldots, m$. We consider a spectral norm bound on the magnitude of the perturbations such that $\|\mathbf{blkdiag}(\Delta_1, \ldots, \Delta_m)\| \leq \rho$.

Incorporating the perturbations, the E-optimal experiment design problem with uncertainty based on (4) can be cast as the following minimax problem:

$$\mathcal{P}_\rho : p_\rho^* = \quad \min_{w,s} \max_{\|\Delta\| \leq \rho} -s$$
$$\text{subject to} \quad \sum_{i=1}^m w_i(v_i v_i^T - \Delta_i) \geq sI_p$$
$$\Delta = \mathbf{blkdiag}(\Delta_1, \ldots, \Delta_m) \tag{5}$$
$$\sum_{i=1}^m w_i = 1, \quad w_i \geq 0, \forall i.$$

We will call equation (5) an *E-robust experiment design*.

To implement the program efficiently, we can recast the linear matrix inequality in (5) in a linear fractional representation:

$$F(w, s, \Delta) = F(w, s) + L\Delta R(w) + R(w)^T \Delta^T L^T \geq 0,$$

where

$$F(w, s) = \sum_{i=1}^{m} w_i v_i v_i^T - sI_p, \qquad R(w) = \tfrac{1}{\sqrt{2}} \left( w \otimes I_p \right)$$

$$L = \frac{-1}{\sqrt{2}} \left( \mathbf{1}_m^T \otimes I_p \right), \quad \Delta = \mathbf{blkdiag}(\Delta_1, \ldots, \Delta_m).$$

Taking $\Delta_1 = \cdots = \Delta_m$, a special case of the S-procedure [11] yields the following semidefinite program:

$$\mathcal{P}_\rho : p_\rho^* = \min_{w,s,\tau} -s$$

$$\text{subject to} \quad \begin{bmatrix} \sum_{i=1}^{m} w_i v_i v_i^T - sI_p - \frac{m}{2}\tau I_p & w^T \otimes \frac{\rho}{\sqrt{2}} I_p \\ w \otimes \frac{\rho}{\sqrt{2}} I_p & \tau I_{mp} \end{bmatrix} \geq 0 \qquad (7)$$

$$\sum_{i=1}^{m} w_i = 1, \quad w_i \geq 0, \forall i.$$

If $\rho = 0$ we recover (4). Using the Schur complement the first constraint in (7) can be further simplified to

$$\sum_{i=1}^{m} w_i v_i v_i^T - \rho\sqrt{m}\|w\|_2 \geq sI_p, \qquad (8)$$

which makes the regularization of the optimization problem (4) explicit. The uncertainty bound, $\rho$, serves as a weighting parameter for a Tikhonov regularization term.

# 5  Results

We demonstrate the robust experiment design on two models of biological systems. The first model is the Michaelis-Menten model of a simple enzyme reaction system. This model, derived from mass-action kinetics, is a fundamental building block of many mechanistic models of biological systems. The second example is a model of a complex calcium signal transduction pathway in macrophage immune cells. In this example we consider RNAi knockdowns at a variety of ligand doses for the estimation of receptor level parameters.

## 5.1  Michaelis-Menten Reaction Model

The Michaelis-Menten model is a common approximation to an enzyme-substrate reaction [13]. The basic chemical reaction that leads to this model is $E + S \underset{k_{-1}}{\overset{k_{+1}}{\rightleftharpoons}} C \xrightarrow{k_2} E + P$, where $E$ is the enzyme concentration, $S$ is the substrate concentration and $P$ is the product concentration. We employ mass action kinetics to develop a differential equation model for this reaction system [13]. The velocity of the reaction is defined to be the rate of product formation, $V_0 = \frac{\partial P}{\partial t}\big|_{t_0}$. The initial velocity of the reaction is

$$V_0 \approx \frac{\theta_1 x}{\theta_2 + x}, \qquad (9)$$

where

$$\theta_1 = k_{+2}E_0, \quad \theta_2 = \frac{k_{-1} + k_{+2}}{k_{+1}}. \qquad (10)$$

We have taken the controllable factor, $x$, in this system to be the initial substrate concentration $S_0$. The parameter $\theta_1$ is the saturating velocity and $\theta_2$ is the initial substrate concentration at which product is formed at one-half the maximal velocity. In this example $\theta_1 = 2$ and $\theta_2 = 2$ are the total enzyme and initial substrate concentrations. We consider six initial substrate concentrations as the menu of experiments, $\mathcal{X} = \left\{ \frac{1}{8}, 1, 2, 4, 8, 16 \right\}$.

Figure 1 shows the robust experiment design weights as a function of the uncertainty parameter with the Jacobian computed at the true parameter values. When $\rho$ is small, the experimental weight is concentrated on only two design points. As $\rho \to \rho_{\max}$ the design converges to a uniform distribution over the entire menu of design points. In a sense, this uniform allocation of experimental energy is most robust to parameter uncertainty. Intermediate values of $\rho$ yield an allocation of design points that reflects a tradeoff between robustness and nominal optimality.

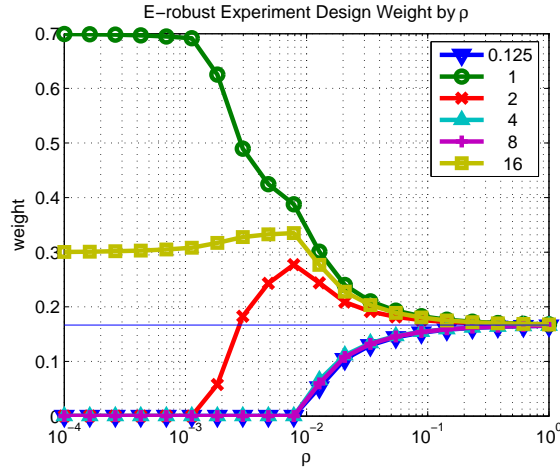

Figure 1: Michaelis-Menten model experiment design weights as a function of $\rho$.

For moderate values of $\rho$ we gain significantly in terms of robustness to errors in $v_i v_i^T$, at a moderate cost to maximal value of the minimum eigenvalues of the parameter estimate covariance matrix. Figure 2 shows the efficiency of the experiment design as a function of $\rho$ and the prior estimate $\theta_{02}$ used to compute the Jacobian matrix. The E-efficiency of a design is defined to be

$$\text{efficiency} \triangleq \frac{\lambda_{\max}\left[\text{cov}\left(\hat{\theta}|\theta, \xi_0\right)\right]}{\lambda_{\max}\left[\text{cov}\left(\hat{\theta}|\theta_0, \xi_\rho\right)\right]}. \tag{11}$$

If the Jacobian is computed at the correct point in parameter space the optimal design achieves maximal efficiency. As the distance between $\theta_0$ and $\theta$ grows the efficiency of the optimal design decreases rapidly. If the estimate, $\theta_{02}$, is eight instead of the true value, two, the efficiency of the optimal design at $\theta_0$ is 36% of the optimal design at $\theta$. However, at the cost of a decrease in efficiency for parameter estimates close to the true parameter value we guarantee the efficiency is better for points further from the true parameters with a robust design. For example, for $\rho = 0.001$ the robust design is less efficient for the range $0 < \theta_{02} < 7$, but is more efficient for $7 < \theta_{02} < 16$.

## 5.2 Calcium Signal Transduction Model

When certain small molecule ligands such as the anaphylatoxin C5a are introduced into the environment of an immune cell a complex chain of chemical reactions leads to the

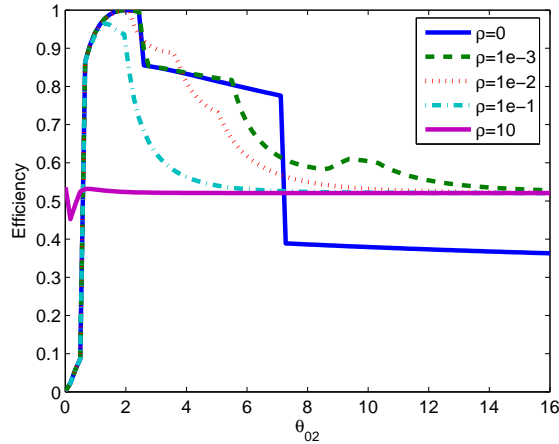

Figure 2: Efficiency of robust designs as a function of $\rho$ and perturbations in the prior parameter estimate $\theta_{02}$.

transduction of the extracellular ligand concentration information and a transient increase in the intracellular calcium concentration. This chain of reactions can be mathematically modeled using the principles of mass-action kinetics and nonlinear ordinary differential equations. We consider specifically the model presented in [14] which was developed for the P2Y2 receptor, modifying the model for our data on the C5a receptor.

The menu of available experiments is indexed by one of two different cell lines in combination with different ligand doses. The cell lines are: wild-type and a *GRK2* knockdown line. *GRK2* is a protein that represses signaling in the G-protein receptor complex. When its concentration is decreased with interfering RNA the repression of the signal due to *GRK2* is reduced. There are 17 experiments on the menu and we choose to do 100 experiments allocated according the experiment design. For each experiment we are able measure the transient calcium spike peak height using a fluorescent calcium dye. We are concerned with estimating three C5A receptor parameters: $K_1$, $k_p$, $k_{deg}$ which are detailed in [14]. We have selected the initial parameter estimates based on a least-squares fit to a separate data set of 67 experiments on a wild-type cell line with a ligand concentration of 250nM. We have estimated, from experimental data, the mean and variance for all of the experiments in our menu. Observations are simulated from these data to obtain the least-squares parameter estimate for the optimal, robust ($\rho = 1.5 \times 10^{-6}$) and uniform experiment designs.

Figure 3 shows the model fits with associated 95% confidence bands for the wild-type and knockdown cell lines for the parameter estimates from the three experiment designs. A separate validation data set is generated uniformly across the design menu. Compared to the optimal design, the parameter estimates based on the robust design provide a better fit across the whole dose range for both cell types as measured by mean-squared residual error.

Note also that the measured response at high ligand concentration is better fit with parameters estimated from the robust design. Near 1$\mu$M of C5a concentration the peak height is predicted to decrease slightly in the wild-type cell line, but plateaus for the *GRK2* knockdown cell line. This matches the biochemical understanding that *GRK2* acts as a repressor of signaling.

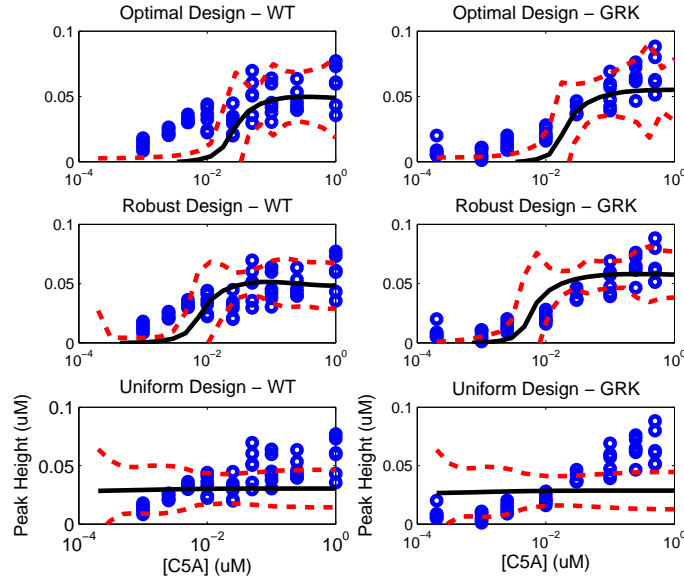

Figure 3: Model predictions based on the least squares parameter estimate using data observed from the optimal, robust and uniform design. The predicted peak height curve (black line) based on the robust design data is shifted to the left compared to the peak height curve based on the optimal design data and matches the validation sample (shown as blue dots) more accurately.

## 6 Discussion

The methodology of optimal experiment design leads to efficient algorithms for the construction of designs in general nonlinear situations [15]. However, these variance-minimizing designs fail to account for uncertainty in the nominal parameter estimate and the model. We present a methodology, based on recent advances in semidefinite programming, that retains the advantages of the general purpose algorithm while explicitly incorporating uncertainty.

We demonstrated this robust experiment design method on two example systems. In the Michaelis-Menten model, we showed that the E-optimal design is recovered for $\rho = 0$ and the uniform design is recovered as $\rho \to \rho_{\max}$. It was also shown that the robust design is more efficient than the optimal for large perturbations of the nominal parameter estimate away from the true parameter.

The second example, of a calcium signal transduction model, is a more realistic case of the need for experiment design in high-throughput biological research. The model captures some of the important kinetics of the system, but is far from complete. We require a reasonably accurate model to make further predictions about the system and drive a set of experiments to estimate critical parameters of the model more accurately. The resulting robust design spreads some experiments across the menu, but also concentrates on experiments that will help minimize the variance of the parameter estimates.

These robust experiment designs were obtained using SeDuMi 1.05 [16]. The design for the calcium signal transduction model takes approximately one second on a 2GHz processor, which is less time than required to compute the Jacobian matrix for the model.

Research in machine learning has led to significant advances in computationally-efficient

data analysis methods, allowing increasingly complex models to be fit to biological data. Challenges in experimental design are the flip side of this coin—for complex models to be useful in closing the loop in biological research it is essential to begin to focus on the development of computationally-efficient experimental design methods.

## Acknowledgments

We would like to thank Andy Packard for helpful discussions. We would also like to thank Robert Rebres and William Seaman for the data used in the second example. PF and APA would like to acknowledge support from the Howard Hughes Medical Institute and from the Alliance for Cellular Signaling through the NIH Grant Number 5U54 GM62114-05. MIJ would like to thank NIH R33 HG003070 for funding.

## References

[1] I. Ford, D.M. Titterington, and C.P. Kitsos. Recent advances in nonlinear experiment design. *Technometrics*, 31(1):49–60, 1989.

[2] L. Vandenberghe, S. Boyd, and W. S.-P. Determinant maximization with linear matrix inequality constraints. *SIAM Journal on Matrix Analysis and Applications*, 19(2):499–533, 1998.

[3] G.A.F. Seber and C.J. Wild. *Nonlinear Regression*. Wiley-Interscience, Hoboken, NJ, 2003.

[4] A.C. Atkinson and A.N. Donev. *Optimum Experimental Designs*. Oxford University Press, 1992.

[5] S. Boyd and L. Vandenberghe. *Convex Optimization*. Cambridge University Press, 2003.

[6] G.E.P Box, W.G. Hunter, and J.S. Hunter. *Statistics for Experimenters: An Introduction to Design, Data Analysis, and Model Building*. John Wiley and Sons, New York, 1978.

[7] S.D. Silvey. *Optimal Design*. Chapman and Hall, London, 1980.

[8] D.V. Lindley. On the measure of information provided by an experiment. *The Annals of Mathematical Statistics*, 27(4):986–1005, 1956.

[9] L. Pronzato and E. Walter. Robust experiment design via maximin optimization. *Mathematical Biosciences*, 89:161–176, 1988.

[10] L. Vandenberghe and S. Boyd. Semidefinite programming. *SIAM Review*, 38(1):49–95, 1996.

[11] L. El Ghaoui, L. Oustry, and H. Lebret. Robust solutions to uncertain semidefinite programs. *SIAM J. Optimization*, 9(1):33–52, 1998.

[12] L. El Ghaoui and H. Lebret. Robust solutions to least squares problems with uncertain data. *SIAM J. Matrix Anal. Appl.*, 18(4):1035–1064, 1997.

[13] L.A. Segel and M. Slemrod. The quasi-steady state assumption: A case study in perturbation. *SIAM Review*, 31(3):446–477, 1989.

[14] G. Lemon, W.G. Gibson, and M.R. Bennett. Metabotropic receptor activation, desensitization and sequestrationi: modelling calcium and inositol 1,4,5-trisphosphate dynamics following receptor activation. *Journal of Theoretical Biology*, 223(1):93–111, 2003.

[15] A.C. Atkinson. The usefulness of optimum experiment designs. *JRSS B*, 58(1):59–76, 1996.

[16] J.F. Sturm. Using SeDuMi 1.02, a MATLAB toolbox for optimization over symmetric cones. *Optimization Methods and Software*, 11:625–653, 1999.
